# Facial Memory is Kernel Density Estimation (Almost)

**Matthew N. Dailey    Garrison W. Cottrell**
Department of Computer Science and Engineering
U.C. San Diego
La Jolla, CA 92093-0114
{mdailey,gary}@cs.ucsd.edu

**Thomas A. Busey**
Department of Psychology
Indiana University
Bloomington, IN 47405
busey@indiana.edu

## Abstract

We compare the ability of three exemplar-based memory models, each using three different face stimulus representations, to account for the probability a human subject responded "old" in an old/new facial memory experiment. The models are 1) the Generalized Context Model, 2) SimSample, a probabilistic sampling model, and 3) MMOM, a novel model related to kernel density estimation that explicitly encodes stimulus distinctiveness. The representations are 1) positions of stimuli in MDS "face space," 2) projections of test faces onto the "eigenfaces" of the study set, and 3) a representation based on response to a grid of Gabor filter jets. Of the 9 model/representation combinations, only the distinctiveness model in MDS space predicts the observed "morph familiarity inversion" effect, in which the subjects' false alarm rate for morphs between similar faces is higher than their hit rate for many of the studied faces. This evidence is consistent with the hypothesis that human memory for faces is a kernel density estimation task, with the caveat that distinctive faces require larger kernels than do typical faces.

## 1  Background

Studying the errors subjects make during face recognition memory tasks aids our understanding of the mechanisms and representations underlying memory, face processing, and visual perception. One way of evoking such errors is by testing subjects' recognition of new faces created from studied faces that have been combined in some way (e.g. Solso and McCarthy, 1981; Reinitz, Lammers, and Cochran 1992). Busey and Tunnicliff (submitted) have recently examined the extent to which image-quality morphs between unfamiliar faces affect subjects' tendency to make recognition errors.

Their experiments used facial images of bald males and morphs between these images (see

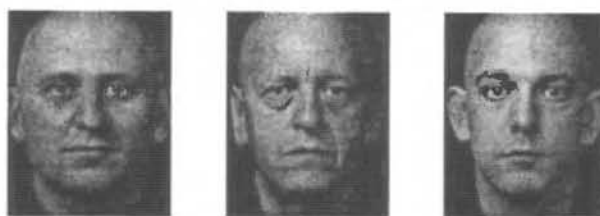

Figure 1: Three normalized morphs from the database.

Figure 1) as stimuli. In one study, Busey (in press) had subjects rate the similarity of all pairs in a large set of faces and morphs, then performed a multidimensional scaling (MDS) of these similarity ratings to derive a 6-dimensional "face space" (Valentine and Endo, 1992). In another study, "Experiment 3" (Busey and Tunnicliff, submitted), 179 subjects studied 68 facial images, including 8 *similar* pairs and 8 *dissimilar pairs*, as determined in a pilot study. These pairs were included in order to study how morphs between similar faces and dissimilar faces evoke false alarms. We call the pair of images from which a morph are derived its "parents," and the morph itself as their "child." In the experiment's test phase, the subjects were asked to make new/old judgments in response to 8 of the 16 morphs, 20 completely new distractor faces, the 36 non-parent targets and one of the parents of each of the 8 morphs. The results were that, for many of the morph/parent pairs, subjects responded "old" to the unstudied morph more often than to its studied parent. However, this effect (a *morph familiarity inversion*) only occurred for the morphs with *similar* parents. It seems that the similar parents are so similar to their "child" morphs that they both contribute toward an "old" (false alarm) response to the morph.

Researchers have proposed many models to account for data from explicit memory experiments. Although we have applied other types of models to Busey and Tunnicliff's data with largely negative results (Dailey et al., 1998), in this paper, we limit discussion to *exemplar-based* models, such as the Generalized Context Model (Nosofsky, 1986) and SAM (Gillund and Shiffrin, 1984). These models rely on the assumption that subjects explicitly store representations of each of the stimuli they study. Busey and Tunnicliff applied several exemplar-based models to the Experiment 3 data, but none of these models have been able to fully account for the observed similar morph familiarity inversion without positing that the similar parents are explicitly blended in memory, producing prototypes near the morphs.

We extend Busey and Tunnicliff's (submitted) work by applying two of their exemplar models to additional image-based face stimulus representations, and we propose a novel exemplar model that accounts for the similar morphs' familiarity inversion. The results are consistent with the hypothesis that facial memory is a kernel density estimation (Bishop, 1995) task, except that *distinctive exemplars require larger kernels*. Also, on the basis of our model, we can predict that distinctiveness *with respect to the study set* is the critical factor influencing kernel size, as opposed to a context-free notion of distinctiveness. We can easily test this prediction empirically.

## 2 Experimental Methods

### 2.1 Face Stimuli and Normalization

The original images were 104 digitized 560x662 grayscale images of bald men, with consistent lighting and background and fairly consistent position. The subjects varied in race and extent of facial hair. We automatically located the left and right eyes on each face using a simple template correlation technique then translated, rotated, scaled and cropped each image so the eyes were aligned in each image. We then scaled each image to 114x143 to speed up image processing. Figure 1 shows three examples of the normalized morphs (the original images are copyrighted and cannot be published).

## 2.2  Representations

**Positions in multidimensional face space**   Many researchers have used a multidimensional scaling approach to model various phenomena in face processing (e.g. Valentine and Endo, 1992). Busey (in press) had 343 subjects rate the similarity of pairs of faces in the test set and performed a multidimensional scaling on the similarity matrix for 100 of the faces (four non-parent target faces were dropped from this analysis). The process resulted in a 6-dimensional solution with $r^2 = 0.785$ and a stress of 0.13. In the MDS modeling results described below, we used the 6-dimensional vector associated with each stimulus as its representation.

**Principal component projections**   "Eigenfaces," or the eigenvectors of the covariance matrix for a set of face images, are a common basis for face representations (e.g. Turk and Pentland, 1991). We performed a principal components analysis on the 68 face images used in the study set for Busey and Tunnicliff's experiment to get the 67 non-zero eigenvectors of their covariance matrix. We then projected each of the 104 test set images onto the 30 most significant eigenfaces to obtain a 30-dimensional vector representing each face.[1]

**Gabor filter responses**   von der Malsburg and colleagues have made effective use of banks of Gabor filters at various orientations and spatial frequencies in face recognition systems. We used one form of their wavelet (Buhmann, Lades, and von der Malsburg, 1990) at five scales and 8 orientations in an 8x8 square grid over each normalized face image as the basis for a third face stimulus representation. However, since this representation resulted in a 2560-dimensional vector for each face stimulus, we performed a principal components analysis to reduce the dimensionality to 30, keeping this representation's dimensionality the same as the eigenface representation's. Thus we obtained a 30-dimensional vector based on Gabor filter responses to represent each test set face image.

## 2.3  Models

**The Generalized Context Model (GCM)**   There are several different flavors to the GCM. We only consider a simple sum-similarity form that will lead directly to our distinctiveness-modulated density estimation model. Our version of GCM's predicted P(old), given a representation $\mathbf{y}$ of a test stimulus and representations $\mathbf{x} \in \mathbf{X}$ of the studied exemplars, is

$$pred_{\mathbf{y}} = \alpha + \beta \sum_{\mathbf{x} \in \mathbf{X}} e^{-c(d_{\mathbf{x},\mathbf{y}})^2}$$

where $\alpha$ and $\beta$ linearly convert the probe's summed similarity to a probability, $\mathbf{X}$ is the set of representations of the study set stimuli; $c$ is used to widen or narrow the width of the similarity function, and $d_{\mathbf{x},\mathbf{y}}$ is either $\|\mathbf{x} - \mathbf{y}\|$, the Euclidean distance between $\mathbf{x}$ and $\mathbf{y}$ or the weighted Euclidean distance $\sqrt{\sum_k w_k (x_k - y_k)^2}$ where the "attentional weights" $w_k$ are constants that sum to 1. Intuitively, this model simply places a Gaussian-shaped function over each of the studied exemplars, and the predicted familiarity of a test probe is simply the summed height of each of these surfaces at the probe's location.

Recall that two of our representations, PC projection space and Gabor filter space, are 30-dimensional, whereas the other, MDS, is only 6-dimensional. Thus allowing adaptive weights for the MDS representation is reasonable, since the resulting model only uses 8 parameters to fit 100 points, but it is clearly unreasonable to allow adaptive weights in PC and Gabor space, where the resulting models would be fitting 32 parameters to 100 points. Thus, for all models, we report results in MDS space both with and without adaptive weights, but do not report adaptive weight results for models in PC and Gabor space.

**SimSample**   Busey and Tunnicliff (submitted) proposed SimSample in an attempt to remedy the GCM's poor predictions of the human data. It is related to both GCM, in that it

uses representations in MDS space, and SAM (Gillund and Shiffrin, 1984), in that it involves sampling exemplars. The idea behind the model is that when a subject is shown a test stimulus, instead of a summed comparison to all of the exemplars in memory, the test probe probabilistically samples a *single* exemplar in memory, and the subject responds "old" if the probe's similarity to the exemplar is above a noisy criterion. The model has a similarity scaling parameter and two parameters describing the noisy threshold function. Due to space limitations, we cannot provide the details of the model here.

Busey and Tunnicliff were able to fit the human data within the SimSample framework, but only when they introduced prototypes at the locations of the morphs in MDS space and made the probability of sampling the prototype proportional to the similarity of the parents. Here, however, we only compare with the basic version that does not blend exemplars.

**Mixture Model of Memory (MMOM)** In this model, we assume that subjects, at study time, implicitly create a probability density surface corresponding to the training set. The subjects' probability of responding "old" to a probe are then taken to be proportional to the height of this surface at the point corresponding to the probe. The surface must be robust in the face of the variability or noise typically encountered in face recognition (lighting changes, perspective changes, etc.) yet also provide some level of discrimination support (i.e. even when the intervals of possible representations for a single face could overlap due to noise, some rational decision boundary must still be constructed). If we assume a Gaussian mixture model, in which the density surface is built from Gaussian "blobs" centered on each studied exemplar, the task is a form of kernel density estimation (Bishop, 1995).

We can formulate the task of predicting the human subjects' P(old) in this framework, then, as optimizing the priors and widths of the kernel functions to minimize the mean squared error of the prediction. However, we also want to minimize the number of free parameters in the model — parsimonious methods for setting the priors and kernel function widths potentially lead to more useful insights into the principles underlying the human data. If the priors and widths were held constant, we would have a simple two parameter model predicting the probability a subject responds "old" to a test stimulus $\mathbf{y}$:

$$pred_{\mathbf{y}} = \sum_{\mathbf{x} \in X} \alpha e^{-\frac{\|\mathbf{x}-\mathbf{y}\|^2}{2\sigma^2}}$$

where $\alpha$ folds together the uniform prior and normalization constants, and $\sigma$ is the standard deviation of the Gaussian kernels. If we ignore the constants, however, this model is essentially the same as the version of the GCM described above. As the results section will show, this model cannot fully account for the human familiarity data in any of our representational spaces.

To improve the model, we introduce two parameters to allow the prior (kernel function height) and standard deviation (kernel function width) to vary with the *distinctiveness* of the studied exemplar. This modification has two intuitive motivations. First, when humans are asked which of two parent faces a 50% morph is most similar to, if one parent is distinctive and the other parent is typical, subjects tend to choose the more distinctive parent (Tanaka et al., submitted). Second, we hypothesize that when a human is asked to study and remember a set of faces for a recognition test, faces with few neighbors will likely have more relaxed (wider) discrimination boundaries than faces with many nearby neighbors.

Thus in each representation space, for each studied face $\mathbf{x}$, we computed $d(\mathbf{x})$, the theoretical distinctiveness of each face, as the Z-scored average distance to the five nearest studied faces. We then allowed the height and width of each kernel function to vary with $d(\mathbf{x})$:

$$pred_{\mathbf{y}} = \sum_{\mathbf{x} \in X} \alpha(1 + c_\alpha d(\mathbf{x}))e^{-\frac{\|\mathbf{x}-\mathbf{y}\|^2}{2(\sigma(1+c_\sigma d(\mathbf{x}))^2}}$$

As was the case for GCM and SimSample, we report the results of using a weighted Euclidean distance between $\mathbf{y}$ and $\mathbf{x}$ in MDS space only.

| Model | MDS space | MDS + weights | PC projections | Gabor jets |
|-------|-----------|---------------|----------------|------------|
| GCM | 0.1633 | 0.1417 | 0.1745 | 0.1624 |
| SimSample | 0.1521 | 0.1404 | 0.1756 | 0.1704 |
| MMOM | 0.1601 | 0.1528 | 0.1992 | 0.1668 |

Table 1: RMSE for the three models and three representations. Quality of fit for models with adaptive attentional weights are only reported for the low-dimensional representation ("MDS + weights"). The baseline RMSE, achievable with a constant prediction, is 0.2044.

## 2.4   Parameter fitting and model evaluation

For each of the twelve combinations of models with face representations, we searched parameter space by simple hill climbing for the parameter settings that minimized the mean squared error between the model's predicted P(old) and the actual human P(old) data.

We rate each model's effectiveness with two criteria. First, we measure the models' global fit with RMSE over all test set points. A model's RMSE can be compared to the baseline performance of the "dumbest" model, which simply predicts the mean human P(old) of 0.5395, and achieves an RMSE of 0.2044. Second, we evaluate the extent to which a model predicts the mean human response for each of the six categories of test set stimuli: 1) non-parent targets, 2) non-morph distractors, 3) similar parents, 4) dissimilar parents, 5) similar morphs, and 6) dissimilar morphs. If a model correctly predicts the rank ordering of these category means, it obviously accounts for the similar morph familiarity inversion pattern in the human data. As long as models do an adequate job of fitting the human data overall, as measured by RMSE, we prefer models that predict the morph familiarity inversion effect as a natural consequence of minimizing RMSE.

## 3   Results

Table 1 shows the global fit of each model/representation pair. The SimSample model in MDS space provides the best quantitative fit. GCM generally outperforms MMOM, indicating that for a tight quantitative fit, having parameters for a linear transformation built into the model is more important than allowing the kernel function to vary with distinctiveness. Also of note is that the PC projection representation is consistently outperformed by both the Gabor jet representation and the MDS space representation.

But for our purposes, the degree to which a model predicts the mean human responses for each of the six categories of stimuli is more important, given that it is doing a reasonably good job globally. Figure 2 takes a more detailed look at how well each model predicts the human category means. Even though SimSample in MDS space has the best global fit to the human familiarity ratings, it does not predict the familiarity inversion for similar morphs. Only the mixture model in weighted MDS space correctly predicts the morph familiarity effect. All of the other models underpredict the human responses to the similar morphs.

## 4   Discussion

The results for the mixture model are consistent with the hypothesis that facial memory is a kernel density estimation task, with the caveat that distinctive exemplars require larger kernels. Whereas true density estimation would tend to deemphasize outliers in sparse areas of the face space, the human data show that the priors and kernel function widths for outliers should actually be increased. Two potentially significant problems with the work presented here are first, we experimented with several models before finding that MMOM was able to predict the morph familiarity inversion effect, and second, we are fitting a single

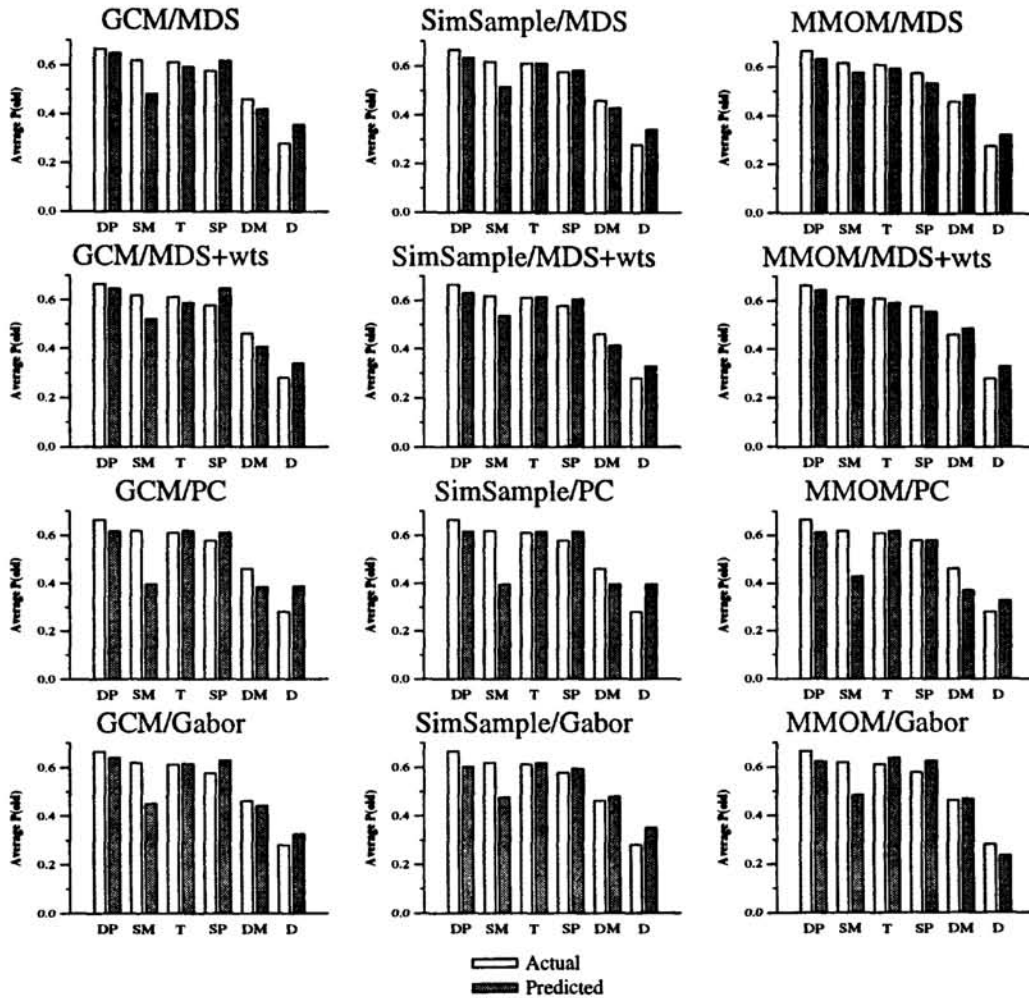

Figure 2: Average actual/predicted responses to the faces in each category. Key: DP = Dissimilar parents; SM = Similar morphs; T = Non-parent targets; SP = Similar parents; DM = Dissimilar morphs; D = Distractors.

experiment. The model thus must be carefully tested against new data, and its predictions empirically validated.

Since a theoretical distinctiveness measure based on the sparseness of face space around an exemplar was sufficient to account for the similar morphs' familiarity inversion, we predict that distinctiveness with respect to the study set is the critical factor influencing kernel size, rather than context-free human distinctiveness judgments. We can easily test this prediction by having subjects rate the distinctiveness of the stimuli without prior exposure and then determine whether their distinctiveness ratings improve or degrade the model's fit.

A somewhat disappointing (though not particularly surprising) aspect of our results is that the model requires a representation based on human similarity judgments. Ideally, we would prefer to provide an information-processing account using image-based representations like eigenface projections or Gabor filter responses. Interestingly, the efficacy of the image-based representations seems to depend on how similar they are to the MDS representations. The PC projection representation performed the worst, and distances between pairs of PC representations had a correlation of 0.388 with the distances between pairs of MDS representations. For the Gabor filter representation, which performed better, the correlation is 0.517. In future work, we plan to investigate how the MDS representation (or a representation like it) might be derived directly from the face images.

Besides providing an information-processing account of the human data, there are several other avenues for future research. These include empirical testing of our distinctiveness predictions, evaluating the applicability of the distinctiveness model in domains other than face processing, and evaluating the ability of other modeling paradigms to account for this data.

## Acknowledgements

We thank Chris Vogt for comments on a previous draft, and other members of Gary's Unbelievable Research Unit (GURU) for earlier comments on this work. This research was supported in part by NIMH grant MH57075 to GWC.

## Footnotes

[1]We used 30 eigenfaces because with this number, our theoretical "distinctiveness" measure was best correlated with the same measure in MDS space.

## References

Bishop, C. M. (1995). *Neural networks for pattern recognition.* Oxford University Press, Oxford.

Busey, T. A. (1999). Where are morphed faces in multi-dimensional face space? *Psychological Science.* In press.

Busey, T. A. and Tunnicliff, J. (submitted). Accounts of blending, distinctiveness and typicality in face recognition. *Journal of Experimental Psychology: Learning, Memory, and Cognition.*

Dailey, M. N., Cottrell, G. W., and Busey, T. A. (1998). Eigenfaces for familiarity. In *Proceedings of the Twentieth Annual Conference of the Cognitive Science Society*, pages 273–278, Mahwah, NJ. Erlbaum.

Gillund, G. and Shiffrin, R. (1984). A retrieval model for both recognition and recall. *Psychological Review*, 93(4):411–428.

J. Buhmann, M. L. and von der Malsburg, C. (1990). Size and distortion invariant object recognition by hierarchical graph matching. In *Proceedings of the IJCNN International Joint Conference on Neural Networks*, volume II, pages 411–416.

Nosofsky, R. M. (1986). Attention, similarity, and the identification-categorization relationship. *Journal of Experimental Psychology: General*, 116(1):39–57.

Reinitz, M., Lammers, W., and Cochran, B. (1992). Memory-conjunction errors: Miscombination of stored stimulus features can produce illusions of memory. *Memory & Cognition*, 20(1):1–11.

Solso, R. L. and McCarthy, J. E. (1981). Prototype formation of faces: A case of pseudomemory. *British Journal of Psychology*, 72(4):499–503.

Tanaka, J., Giles, M., Kremen, S., and Simon, V. (submitted). Mapping attractor fields in face space: The atypicality bias in face recognition.

Turk, M. and Pentland, A. (1991). Eigenfaces for recognition. *The Journal of Cognitive Neuroscience*, 3:71–86.

Valentine, T. and Endo, M. (1992). Towards an exemplar model of face processing: The effects of race and distinctiveness. *The Quarterly Journal of Experimental Psychology*, 44A(4):671–703.